# Nearly Tight Bounds for the Continuum-Armed Bandit Problem

**Robert Kleinberg**[*]

## Abstract

In the multi-armed bandit problem, an online algorithm must choose from a set of strategies in a sequence of $n$ trials so as to minimize the total cost of the chosen strategies. While nearly tight upper and lower bounds are known in the case when the strategy set is finite, much less is known when there is an infinite strategy set. Here we consider the case when the set of strategies is a subset of $\mathbb{R}^d$, and the cost functions are continuous. In the $d = 1$ case, we improve on the best-known upper and lower bounds, closing the gap to a sublogarithmic factor. We also consider the case where $d > 1$ and the cost functions are convex, adapting a recent online convex optimization algorithm of Zinkevich to the sparser feedback model of the multi-armed bandit problem.

## 1 Introduction

In an online decision problem, an algorithm must choose from among a set of strategies in each of $n$ consecutive trials so as to minimize the total cost of the chosen strategies. The costs of strategies are specified by a real-valued function which is defined on the entire strategy set and which varies over time in a manner initially unknown to the algorithm. The archetypical online decision problems are the *best expert problem*, in which the entire cost function is revealed to the algorithm as feedback at the end of each trial, and the *multi-armed bandit problem*, in which the feedback reveals only the cost of the chosen strategy. The names of the two problems are derived from the metaphors of combining expert advice (in the case of the best expert problem) and learning to play the best slot machine in a casino (in the case of the multi-armed bandit problem).

The applications of online decision problems are too numerous to be listed here. In addition to occupying a central position in online learning theory, algorithms for such problems have been applied in numerous other areas of computer science, such as paging and caching [6, 14], data structures [7], routing [4, 5], wireless networks [19], and online auction mechanisms [8, 15]. Algorithms for online decision problems are also applied in a broad range of fields outside computer science, including statistics (sequential design of experiments [18]), economics (pricing [20]), game theory (adaptive game playing [13]), and medical decision making (optimal design of clinical trials [10]).

Multi-armed bandit problems have been studied quite thoroughly in the case of a finite strategy set, and the performance of the optimal algorithm (as a function of $n$) is known

---

[*]M.I.T. CSAIL, Cambridge, MA 02139. Email: `rdk@csail.mit.edu`. Supported by a Fannie and John Hertz Foundation Fellowship.

up to a constant factor [3, 18]. In contrast, much less is known in the case of an infinite strategy set. In this paper, we consider multi-armed bandit problems with a continuum of strategies, parameterized by one or more real numbers. In other words, we are studying online learning problems in which the learner designates a strategy in each time step by specifying a $d$-tuple of real numbers $(x_1, \ldots, x_d)$; the cost function is then evaluated at $(x_1, \ldots, x_d)$ and this number is reported to the algorithm as feedback. Recent progress on such problems has been spurred by the discovery of new algorithms (e.g. [4, 9, 16, 21]) as well as compelling applications. Two such applications are online auction mechanism design [8, 15], in which the strategy space is an interval of feasible prices, and online oblivious routing [5], in which the strategy space is a flow polytope.

Algorithms for online decisions problems are often evaluated in terms of their *regret*, defined as the difference in expected cost between the sequence of strategies chosen by the algorithm and the best fixed (i.e. not time-varying) strategy. While tight upper and lower bounds on the regret of algorithms for the $K$-armed bandit problem have been known for many years [3, 18], our knowledge of such bounds for continuum-armed bandit problems is much less satisfactory. For a one-dimensional strategy space, the first algorithm with sublinear regret appeared in [1], while the first polynomial lower bound on regret appeared in [15]. For Lipschitz-continuous cost functions (the case introduced in [1]), the best known upper and lower bounds for this problem are currently $O(n^{3/4})$ and $\Omega(n^{1/2})$, respectively [1, 15], leaving as an open question the problem of determining tight bounds for the regret as a function of $n$. Here, we solve this open problem by sharpening the upper and lower bounds to $O(n^{2/3} \log^{1/3}(n))$ and $\Omega(n^{2/3})$, respectively, closing the gap to a sublogarithmic factor. Note that this requires improving the best known algorithm as well as the lower bound technique.

Recently, and independently, Eric Cope [11] considered a class of cost functions obeying a more restrictive condition on the shape of the function near its optimum, and for such functions he obtained a sharper bound on regret than the bound proved here for uniformly locally Lipschitz cost functions. Cope requires that each cost function $C$ achieves its optimum at a unique point $\theta$, and that there exist constants $K_0 > 0$ and $p \geq 1$ such that for all $x$, $|C(x) - C(\theta)| \geq K_0 \|x - \theta\|^p$. For this class of cost functions — which is probably broad enough to capture most cases of practical interest — he proves that the regret of the optimal continuum-armed bandit algorithm is $O(n^{-1/2})$, and that this bound is tight.

For a $d$-dimensional strategy space, any multi-armed bandit algorithm must suffer regret depending exponentially on $d$ unless the cost functions are further constrained. (This is demonstrated by a simple counterexample in which the cost function is identically zero in all but one orthant of $\mathbb{R}^d$, takes a negative value somewhere in that orthant, and does not vary over time.) For the best-expert problem, algorithms whose regret is polynomial in $d$ and sublinear in $n$ are known for the case of cost functions which are constrained to be linear [16] or convex [21]. In the case of linear cost functions, the relevant algorithm has been adapted to the multi-armed bandit setting in [4, 9]. Here we adapt the online convex programming algorithm of [21] to the continuum-armed bandit setting, obtaining the first known algorithm for this problem to achieve regret depending polynomially on $d$ and sublinearly on $n$. A remarkably similar algorithm was discovered independently and simultaneously by Flaxman, Kalai, and McMahan [12]. Their algorithm and analysis are superior to ours, requiring fewer smoothness assumptions on the cost functions and producing a tighter upper bound on regret.

## 2 Terminology and Conventions

We will assume that a strategy set $\mathcal{S} \subseteq \mathbb{R}^d$ is given, and that it is a compact subset of $\mathbb{R}^d$. Time steps will be denoted by the numbers $\{1, 2, \ldots, n\}$. For each $t \in \{1, 2, \ldots, n\}$ a cost

function $C_t : \mathcal{S} \to \mathbb{R}$ is given. These cost functions must satisfy a continuity property based on the following definition. A function $f$ is *uniformly locally Lipschitz* with constant $L$ $(0 \le L < \infty)$, exponent $\alpha$ $(0 < \alpha \le 1)$, and restriction $\delta$ $(\delta > 0)$ if it is the case that for all $u, u' \in \mathcal{S}$ with $\|u - u'\| \le \delta$,

$$|f(u) - f(u')| \le L\|u - u'\|^\alpha.$$

(Here, $\| \cdot \|$ denotes the Euclidean norm on $\mathbb{R}^d$.) The class of all such functions $f$ will be denoted by $ulL(\alpha, L, \delta)$.

We will consider two models which may govern the cost functions. The first of these is identical with the continuum-armed bandit problem considered in [1], except that [1] formulates the problem in terms of maximizing reward rather than minimizing cost. The second model concerns a sequence of cost functions chosen by an oblivious adversary.

**Random** The functions $C_1, \ldots, C_n$ are independent, identically distributed random samples from a probability distribution on functions $C : \mathcal{S} \to \mathbb{R}$. The expected cost function $\bar{C} : \mathcal{S} \to \mathbb{R}$ is defined by $\bar{C}(u) = \mathbf{E}(C(u))$ where $C$ is a random sample from this distribution. This function $\bar{C}$ is required to belong to $ulL(\alpha, L, \delta)$ for some specified $\alpha, L, \delta$. In addition, we assume there exist positive constants $\zeta, s_0$ such that if $C$ is a random sample from the given distribution on cost functions, then
$$\mathbf{E}(e^{sC(u)}) \le e^{\frac{1}{2}\zeta^2 s^2} \quad \forall |s| \le s_0, u \in \mathcal{S}.$$
The "best strategy" $u^*$ is defined to be any element of $\arg\min_{u \in \mathcal{S}} \bar{C}(u)$. (This set is non-empty, by the compactness of $\mathcal{S}$.)

**Adversarial** The functions $C_1, \ldots, C_n$ are a fixed sequence of functions in $ulL(\alpha, L, \delta)$, taking values in $[0, 1]$. The "best strategy" $u^*$ is defined to be any element of $\arg\min_{u \in \mathcal{S}} \sum_{t=1}^{n} C_t(u)$. (Again, this set is non-empty by compactness.)

A multi-armed bandit algorithm is a rule for deciding which strategy to play at time $t$, given the outcomes of the first $t - 1$ trials. More formally, a deterministic multi-armed bandit algorithm $U$ is a sequence of functions $U_1, U_2, \ldots$ such that $U_t : (\mathcal{S} \times \mathbb{R})^{t-1} \to \mathcal{S}$. The interpretation is that $U_t(u_1, x_1, u_2, x_2, \ldots, u_{t-1}, x_{t-1})$ defines the strategy to be chosen at time $t$ if the algorithm's first $t - 1$ choices were $u_1, \ldots, u_{t-1}$ respectively, and their costs were $x_1, \ldots, x_{t-1}$ respectively. A randomized multi-armed bandit algorithm is a probability distribution over deterministic multi-armed bandit algorithms. (If the cost functions are random, we will assume their randomness is independent of the algorithm's random choices.) For a randomized multi-armed bandit algorithm, the *n-step regret* $R_n$ is the expected difference in total cost between the algorithm's chosen strategies $u_1, u_2, \ldots, u_n$ and the best strategy $u^*$, i.e.

$$R_n = \mathbf{E}\left[\sum_{t=1}^{n} C_t(u_t) - C_t(u^*)\right].$$

Here, the expectation is over the algorithm's random choices and (in the random-costs model) the randomness of the cost functions.

## 3  Algorithms for the one-parameter case ($d = 1$)

The continuum-bandit algorithm presented in [1] is based on computing an estimate $\hat{C}$ of the expected cost function $\bar{C}$ which converges almost surely to $\bar{C}$ as $n \to \infty$. This estimate is obtained by devoting a small fraction of the time steps (tending to zero as $n \to \infty$) to sampling the random cost functions at an approximately equally-spaced sequence of "design points" in the strategy set, and combining these samples using a kernel estimator.

When the algorithm is not sampling a design point, it chooses a strategy which minimizes expected cost according to the current estimate $\hat{C}$. The convergence of $\hat{C}$ to $\bar{C}$ ensures that the average cost in these "exploitation steps" converges to the minimum value of $\bar{C}$.

A drawback of this approach is its emphasis on estimating the entire function $\bar{C}$. Since the algorithm's goal is to minimize cost, its estimate of $\bar{C}$ need only be accurate for strategies where $\bar{C}$ is near its minimum. Elsewhere a crude estimate of $\bar{C}$ would have sufficed, since such strategies may safely be ignored by the algorithm. The algorithm in [1] thus uses its sampling steps inefficiently, focusing too much attention on portions of the strategy interval where an accurate estimate of $\bar{C}$ is unnecessary. We adopt a different approach which eliminates this inefficiency and also leads to a much simpler algorithm. First we discretize the strategy space by constraining the algorithm to choose strategies only from a fixed, finite set of $K$ equally spaced design points $\{1/K, 2/K, \ldots, 1\}$. (For simplicity, we are assuming here and for the rest of this section that $\mathcal{S} = [0, 1]$.) This reduces the continuum-armed bandit problem to a finite-armed bandit problem, and we may apply one of the standard algorithms for such problems. Our continuum-armed bandit algorithm is shown in Figure 1. The outer loop uses a standard doubling technique to transform a non-uniform algorithm to a uniform one. The inner loop requires a subroutine **MAB** which should implement a finite-armed bandit algorithm appropriate for the cost model under consideration. For example, **MAB** could be the algorithm **UCB1** of [2] in the random case, or the algorithm **Exp3** of [3] in the adversarial case. The semantics of **MAB** are as follows: it is initialized with a finite set of strategies; subsequently it recommends strategies in this set, waits to learn the feedback score for its recommendation, and updates its recommendation when the feedback is received.

The analysis of this algorithm will ensure that its choices have low regret relative to the best design point. The Lipschitz regularity of $\bar{C}$ guarantees that the best design point performs nearly as well, on average, as the best strategy in $\mathcal{S}$.

---

ALGORITHM $\mathbf{CAB1}$
$T \leftarrow 1$
**while** $T \leq n$
    $K \leftarrow \left\lceil \left( \frac{T}{\log T} \right)^{\frac{1}{2\alpha+1}} \right\rceil$
    Initialize $\mathbf{MAB}$ with strategy set $\{1/K, 2/K, \ldots, 1\}$.
    **for** $t = T, T+1, \ldots, \min(2T-1, n)$
        Get strategy $u_t$ from $\mathbf{MAB}$.
        Play $u_t$ and discover $C_t(u_t)$.
        Feed $1 - C_t(u_t)$ back to $\mathbf{MAB}$.
    **end**
    $T \leftarrow 2T$
**end**

Figure 1: Algorithm for the one-parameter continuum-armed bandit problem

---

**Theorem 3.1.** *In both the random and adversarial models, the regret of algorithm $\mathbf{CAB1}$ is $O(n^{\frac{\alpha+1}{2\alpha+1}} \log^{\frac{\alpha}{2\alpha+1}}(n))$.*

*Proof Sketch.* Let $q = \frac{\alpha}{2\alpha+1}$, so that the regret bound is $O(n^{1-q} \log^q(n))$. It suffices to prove that the regret in the inner loop is $O(T^{1-q} \log^q(T))$; if so, then we may sum this bound over all iterations of the inner loop to get a geometric progression with constant ratio, whose largest term is $O(n^{1-q} \log^q(n))$. So from now on assume that $T$ is fixed and that $K$ is defined as in Figure 1, and for simplicity renumber the $T$ steps in this iteration of

inner loop so that the first is step 1 and the last is step $T$. Let $u^*$ be the best strategy in $\mathcal{S}$, and let $u'$ be the element of $\{1/K, 2/K, \ldots, 1\}$ nearest to $u^*$. Then $|u' - u^*| < 1/K$, so using the fact that $\bar{C} \in ulL(\alpha, L, \delta)$ (or that $\frac{1}{T} \sum_{t=1}^{T} C_t \in ulL(\alpha, L, \delta)$ in the adversarial case) we obtain

$$\mathbf{E}\left[\sum_{t=1}^{T} C_t(u') - C_t(u^*)\right] \leq \frac{T}{K^\alpha} = O\left(T^{1-q} \log^q(T)\right).$$

It remains to show that $\mathbf{E}\left[\sum_{t=1}^{T} C_t(u_t) - C_t(u')\right] = O\left(T^{1-q} \log^q(T)\right)$. For the adversarial model, this follows directly from Corollary 4.2 in [3], which asserts that the regret of $\mathbf{Exp3}$ is $O\left(\sqrt{TK \log K}\right)$. For the random model, a separate argument is required. (The upper bound for the adversarial model doesn't directly imply an upper bound for the random model, since the cost functions are required to take values in $[0, 1]$ in the adversarial model but not in the random model.) For $u \in \{1/K, 2/K, \ldots, 1\}$ let $\Delta(u) = \bar{C}(u) - \bar{C}(u')$. Let $\Delta = \sqrt{K \log(T)/T}$, and partition the set $\{1/K, 2/K, \ldots, 1\}$ into two subsets $A, B$ according to whether $\Delta(u) < \Delta$ or $\Delta(u) \geq \Delta$. The time steps in which the algorithm chooses strategies in $A$ contribute at most $O(T\Delta) = O(T^{1-q} \log^q(T))$ to the regret. For each strategy $u \in B$, one may prove that, with high probability, $u$ is played only $O(\log(T)/\Delta(u)^2)$ times. (This parallels the corresponding proof in [2] and is omitted here. Our hypothesis on the moment generating function of the random variable $C(u)$ is strong enough to imply the exponential tail inequality required in that proof.) This implies that the time steps in which the algorithm chooses strategies in $B$ contribute at most $O(K \log(T)/\Delta) = O(T^{1-q} \log^q(T))$ to the regret, which completes the proof. $\qquad \square$

## 4 Lower bounds for the one-parameter case

There are many reasons to expect that Algorithm $\mathbf{CAB1}$ is an inefficient algorithm for the continuum-armed bandit problem. Chief among these is that fact that it treats the strategies $\{1/K, 2/K, \ldots, 1\}$ as an unordered set, ignoring the fact that experiments which sample the cost of one strategy $j/K$ are (at least weakly) predictive of the costs of nearby strategies. In this section we prove that, contrary to this intuition, $\mathbf{CAB1}$ is in fact quite close to the optimal algorithm. Specifically, in the regret bound of Theorem 3.1, the exponent of $\frac{\alpha+1}{2\alpha+1}$ is the best possible: for any $\beta < \frac{\alpha+1}{2\alpha+1}$, no algorithm can achieve regret $O(n^\beta)$. This lower bound applies to both the randomized and adversarial models.

The lower bound relies on a function $f : [0, 1] \to [0, 1]$ defined as the sum of a nested family of "bump functions." Let $B$ be a $C^\infty$ bump function defined on the real line, satisfying $0 \leq B(x) \leq 1$ for all $x$, $B(x) = 0$ if $x \leq 0$ or $x \geq 1$, and $B(x) = 1$ if $x \in [1/3, 2/3]$. For an interval $[a, b]$, let $B_{[a,b]}$ denote the bump function $B(\frac{x-a}{b-a})$, i.e. the function $B$ rescaled and shifted so that its support is $[a, b]$ instead of $[0, 1]$. Define a random nested sequence of intervals $[0, 1] = [a_0, b_0] \supset [a_1, b_1] \supset \ldots$ as follows: for $k > 0$, the middle third of $[a_{k-1}, b_{k-1}]$ is subdivided into intervals of width $w_k = 3^{-k!}$, and $[a_k, b_k]$ is one of these subintervals chosen uniformly at random. Now let

$$f(x) = 1/3 + \left(3^{\alpha-1} - 1/3\right) \sum_{k=1}^{\infty} w_k^\alpha B_{[a_k, b_k]}(x).$$

Finally, define a probability distribution on functions $C : [0, 1] \to [0, 1]$ by the following rule: sample $\lambda$ uniformly at random from the open interval $(0, 1)$ and put $C(x) = \lambda^{f(x)}$.

The relevant technical properties of this construction are summarized in the following lemma.

**Lemma 4.1.** *Let $\{u^*\} = \bigcap_{k=1}^{\infty}[a_k, b_k]$. The function $f(x)$ belongs to $ulL(\alpha, L, \delta)$ for some constants $L, \delta$, it takes values in $[1/3, 2/3]$, and it is uniquely maximized at $u^*$. For each $\lambda \in (0,1)$, the function $C(x) = \lambda^{f(x)}$ belongs to $ulL(\alpha, L, \delta)$ for some constants $L, \delta$, and is uniquely minimized at $u^*$. The same two properties are satisfied by the function $\bar{C}(x) = \mathbf{E}_{\lambda \in (0,1)}\left[\lambda^{f(x)}\right] = (1 + f(x))^{-1}$.*

**Theorem 4.2.** *For any randomized multi-armed bandit algorithm, there exists a probability distribution on cost functions such that for all $\beta < \frac{\alpha+1}{2\alpha+1}$, the algorithm's regret $\{R_n\}_{n=1}^{\infty}$ in the random model satisfies*

$$\limsup_{n \to \infty} \frac{R_n}{n^{\beta}} = \infty.$$

*The same lower bound applies in the adversarial model.*

*Proof sketch.* The idea is to prove, using the probabilistic method, that there exists a nested sequence of intervals $[0,1] = [a_0, b_0] \supset [a_1, b_1] \supset \ldots$, such that if we use these intervals to define a probability distribution on cost functions $C(x)$ as above, then $R_n/n^{\beta}$ diverges as $n$ runs through the sequence $n_1, n_2, n_3, \ldots$ defined by $n_k = \lceil \frac{1}{k}(w_{k-1}/w_k)w_k^{-2\alpha} \rceil$. Assume that intervals $[a_0, b_0] \supset \ldots \supset [a_{k-1}, b_{k-1}]$ have already been specified. Subdivide $[a_{k-1}, b_{k-1}]$ into subintervals of width $w_k$, and suppose $[a_k, b_k]$ is chosen uniformly at random from this set of subintervals. For any $u, u' \in [a_{k-1}, b_{k-1}]$, the Kullback-Leibler distance $KL(C(u)\|C(u'))$ between the cost distributions at $u$ and $u'$ is $O(w_k^{2\alpha})$, and it is equal to zero unless at least one of $u, u'$ lies in $[a_k, b_k]$. This means, roughly speaking, that the algorithm must sample strategies in $[a_k, b_k]$ at least $w_k^{-2\alpha}$ times before being able to identify $[a_k, b_k]$ with constant probability. But $[a_k, b_k]$ could be any one of $w_{k-1}/w_k$ possible subintervals, and we don't have enough time to play $w_k^{-2\alpha}$ trials in even a constant fraction of these subintervals before reaching time $n_k$. Therefore, with constant probability, a constant fraction of the strategies chosen up to time $n_k$ are not located in $[a_k, b_k]$, and each of them contributes $\Omega(w_k^{\alpha})$ to the regret. This means the expected regret at time $n_k$ is $\Omega(n_k w_k^{\alpha})$. From this, we obtain the stated lower bound using the fact that

$$n_k w_k^{\alpha} = n_k^{\frac{\alpha+1}{2\alpha+1} - o(1)}.$$

Although this proof sketch rests on a much more complicated construction than the lower bound proof for the finite-armed bandit problem given by Auer et al in [3], one may follow essentially the same series of steps as in their proof to make the sketch given above into a rigorous proof. The only significant technical difference is that we are working with continuous-valued rather than discrete-valued random variables, which necessitates using the differential Kullback-Leibler distance[1] rather than working with the discrete Kullback-Leibler distance as in [3]. □

## 5 An online convex optimization algorithm

We turn now to continuum-armed bandit problems with a strategy space of dimension $d > 1$. As mentioned in the introduction, for any randomized multi-armed bandit algorithm there is a cost function $C$ (with any desired degree of smoothness and boundedness) such that the algorithm's regret is $\Omega(2^d)$ when faced with the input sequence $C_1 = C_2 = \ldots = C_n = C$. As a counterpoint to this negative result, we seek interesting classes of cost functions which admit a continuum-armed bandit algorithm whose regret is polynomial in $d$ (and, as always, sublinear in $n$). A natural candidate is the class of convex, smooth functions on a closed, bounded, convex strategy set $\mathcal{S} \subset \mathbb{R}^d$, since this is the most

general class of functions for which the corresponding best-expert problem is known to admit an efficient algorithm, namely Zinkevich's *greedy projection* algorithm [21]. Greedy projection is initialized with a sequence of learning rates $\eta_1 > \eta_2 > \ldots$. It selects an arbitrary initial strategy $u_1 \in \mathcal{S}$ and updates its strategy in each subsequent time step $t$ according to the rule $u_{t+1} = P(u_t - \eta_t \nabla C_t(u_t))$, where $\nabla C_t(u_t)$ is the gradient of $C_t$ at $u_t$ and $P : \mathbb{R}^d \to \mathcal{S}$ is the projection operator which maps each point of $\mathbb{R}^d$ to the nearest point of $\mathcal{S}$. (Here, distance is measured according to the Euclidean norm.)

Note that greedy projection is nearly a multi-armed bandit algorithm: if the algorithm's feedback when sampling strategy $u_t$ were the vector $\nabla C_t(u_t)$ rather than the number $C_t(u_t)$, it would have all the information required to run greedy projection. To adapt this algorithm to the multi-armed bandit setting, we use the following idea: group the timeline into phases of $d+1$ consecutive steps, with a cost function $C_\phi$ for each phase $\phi$ defined by averaging the cost functions at each time step of $\phi$. In each phase use trials at $d+1$ affinely independent points of $\mathcal{S}$, located at or near $u_t$, to estimate the gradient $\nabla C_\phi(u_t)$.[2]

To describe the algorithm, it helps to assume that the convex set $\mathcal{S}$ is in isotropic position in $\mathbb{R}^d$. (If not, we may bring it into isotropic position by an affine transformation of the coordinate system. This does not increase the regret by a factor of more than $d^2$.) The algorithm, which we will call *simulated greedy projection*, works as follows. It is initialized with a sequence of "learning rates" $\eta_1, \eta_2, \ldots$ and "frame sizes" $\nu_1, \nu_2, \ldots$. At the beginning of a phase $\phi$, we assume the algorithm has determined a *basepoint* strategy $u_\phi$. (An arbitrary $u_\phi$ may be used in the first phase.) The algorithm chooses a set of $(d+1)$ affinely independent points $\{x_0 = u_\phi, x_1, x_2, \ldots, x_d\}$ with the property that for any $y \in \mathcal{S}$, the difference $y - x_0$ may be expressed as a linear combination of the vectors $\{x_i - x_0 : 1 \leq i \leq d\}$ using coefficients in $[-2, 2]$. (Such a set is called an *approximate barycentric spanner*, and may computed efficiently using an algorithm specified in [4].) We then choose a random bijection $\sigma$ mapping the time steps in phase $\phi$ into the set $\{0, 1, \ldots, d\}$, and in step $t$ we sample the strategy $y_t = u_\phi + \nu_\phi(x_{\sigma(t)} - u_\phi)$. At the end of the phase we let $B_\phi$ denote the unique affine function whose values at the points $y_t$ are equal to the costs observed during the phase at those points. The basepoint for the next phase $\phi'$ is determined according to Zinkevich's update rule $u_{\phi'} = P(u_\phi - \eta_\phi \nabla B_\phi(u_\phi))$.[3]

**Theorem 5.1.** *Assume that $\mathcal{S}$ is in isotropic position and that the cost functions satisfy $\|C_t(x)\| \leq 1$ for all $x \in \mathcal{S}, 1 \leq t \leq n$, and that in addition the Hessian matrix of $C_t(x)$ at each point $x \in \mathcal{S}$ has Frobenius norm bounded above by a constant. If $\eta_k = k^{-3/4}$ and $\nu_k = k^{-1/4}$, then the regret of the simulated greedy projection algorithm is $O(d^3 n^{3/4})$.*

*Proof sketch.* In each phase $\phi$, let $Y_\phi = \{y_0, \ldots, y_d\}$ be the set of points which were sampled, and define the following four functions: $C_\phi$, the average of the cost functions in phase $\phi$; $\Lambda_\phi$, the linearization of $C_\phi$ at $u_\phi$, defined by the formula

$$\Lambda_\phi(x) = \nabla C_\phi(u_\phi) \cdot (x - u_\phi) + C_\phi(u_\phi);$$

$L_\phi$, the unique affine function which agrees with $C_\phi$ at each point of $Y_\phi$; and $B_\phi$, the affine function computed by the algorithm at the end of phase $\phi$. The algorithm is simply running greedy projection with respect to the simulated cost functions $B_\phi$, and it consequently satisfies a low-regret bound with respect to those functions. The expected value of $B_\phi(u)$ is $L_\phi(u)$ for every $u$. (Proof: both are affine functions, and they agree on every point of

$Y_\phi$.) Hence we obtain a low-regret bound with respect to $L_\phi$. To transfer this over to a low-regret bound for the original problem, we need to bound several additional terms: the regret experienced because the algorithm was using $u_\phi + \eta_\phi(x_{\sigma(t)} - u_\phi)$ instead of $u_\phi$, the difference between $L_\phi(u^*)$ and $\Lambda_\phi(u^*)$, and the difference between $\Lambda_\phi(u^*)$ and $C_\phi(u^*)$. In each case, the desired upper bound can be inferred from properties of barycentric spanners, or from the convexity of $C_\phi$ and the bounds on its first and second derivatives. $\qquad\square$

## Footnotes

[1] Defined by the formula $KL(P\|Q) = \int \log (p(x)/q(x))\, dp(x)$, for probability distributions $P, Q$ with density functions $p, q$.

[2]Flaxman, Kalai, and McMahan [12], with characteristic elegance, supply an algorithm which counterintuitively obtains an unbiased estimate of the approximate gradient using only a *single* sample. Thus they avoid grouping the timeline into phases and improve the algorithm's convergence time by a factor of $d$.

[3]Readers familiar with Kiefer-Wolfowitz stochastic approximation [17] will note the similarity with our algorithm. The random bijection $\sigma$ —which is unnecessary in the Kiefer-Wolfowitz algorithm —is used here to defend against the oblivious adversary.

## References

[1] R. AGRAWAL. The continuum-armed bandit problem. *SIAM J. Control and Optimization*, 33:1926-1951, 1995.

[2] P. AUER, N. CESA-BIANCHI, AND P. FISCHER. Finite-time analysis of the multi-armed bandit problem. *Machine Learning*, 47:235-256, 2002.

[3] P. AUER, N. CESA-BIANCHI, Y. FREUND, AND R. SCHAPIRE. Gambling in a rigged casino: The adversarial multi-armed bandit problem. In *Proceedings of FOCS 1995*.

[4] B. AWERBUCH AND R. KLEINBERG. Near-Optimal Adaptive Routing: Shortest Paths and Geometric Generalizations. In *Proceedings of STOC 2004*.

[5] N. BANSAL, A. BLUM, S. CHAWLA, AND A. MEYERSON. Online oblivious routing. In *Proceedings of SPAA 2003*: 44-49.

[6] A. BLUM, C. BURCH, AND A. KALAI. Finely-competitive paging. In *Proceedings of FOCS 1999*.

[7] A. BLUM, S. CHAWLA, AND A. KALAI. Static Optimality and Dynamic Search-Optimality in Lists and Trees. *Algorithmica* 36(3): 249-260 (2003).

[8] A. BLUM, V. KUMAR, A. RUDRA, AND F. WU. Online learning in online auctions. In *Proceedings of SODA 2003*.

[9] A. BLUM AND H. B. MCMAHAN. Online geometric optimization in the bandit setting against an adaptive adversary. In *Proceedings of COLT 2004*.

[10] D. BERRY AND L. PEARSON. Optimal Designs for Two-Stage Clinical Trials with Dichotomous Responses. *Statistics in Medicine* 4:487 - 508, 1985.

[11] E. COPE. Regret and Convergence Bounds for Immediate-Reward Reinforcement Learning with Continuous Action Spaces. Preprint, 2004.

[12] A. FLAXMAN, A. KALAI, AND H. B. MCMAHAN. Online Convex Optimization in the Bandit Setting: Gradient Descent Without a Gradient. To appear in *Proceedings of SODA 2005*.

[13] Y. FREUND AND R. SCHAPIRE. Adaptive Game Playing Using Multiplicative Weights. *Games and Economic Behavior* 29:79-103, 1999.

[14] R. GRAMACY, M. WARMUTH, S. BRANDT, AND I. ARI. Adaptive Caching by Refetching. In *Advances in Neural Information Processing Systems 15*, 2003.

[15] R. KLEINBERG AND T. LEIGHTON. The Value of Knowing a Demand Curve: Bounds on Regret for On-Line Posted-Price Auctions. In *Proceedings of FOCS 2003*.

[16] A. KALAI AND S. VEMPALA. Efficient algorithms for the online decision problem. In *Proceedings of COLT 2003*.

[17] J. KIEFER AND J. WOLFOWITZ. Stochastic Estimation of the Maximum of a Regression Function. *Annals of Mathematical Statistics* 23:462-466, 1952.

[18] T. L. LAI AND H. ROBBINS. Asymptotically efficient adaptive allocations rules. *Adv. in Appl. Math.* 6:4-22, 1985.

[19] C. MONTELEONI AND T. JAAKKOLA. Online Learning of Non-stationary Sequences. In *Advances in Neural Information Processing Systems 16*, 2004.

[20] M. ROTHSCHILD. A Two-Armed Bandit Theory of Market Pricing. *Journal of Economic Theory* 9:185-202, 1974.

[21] M. ZINKEVICH. Online Convex Programming and Generalized Infinitesimal Gradient Ascent. In *Proceedings of ICML 2003*, 928-936.
